# Learning Bregman Distance Functions and Its Application for Semi-Supervised Clustering

**Lei Wu**[†♯]**, Rong Jin**[‡]**, Steven C.H. Hoi**[†]**, Jianke Zhu**[♭]**, and Nenghai Yu**[♯]
[†]School of Computer Engineering, Nanyang Technological University, Singapore
[‡]Department of Computer Science & Engineering, Michigan State University
[♭]Computer Vision Lab, ETH Zurich, Swiss
[♯]Univeristy of Science and Technology of China, P.R. China

## Abstract

Learning distance functions with side information plays a key role in many machine learning and data mining applications. Conventional approaches often assume a Mahalanobis distance function. These approaches are limited in two aspects: (i) they are computationally expensive (even infeasible) for high dimensional data because the size of the metric is in the square of dimensionality; (ii) they assume a fixed metric for the entire input space and therefore are unable to handle heterogeneous data. In this paper, we propose a novel scheme that learns *nonlinear* Bregman distance functions from side information using a nonparametric approach that is similar to support vector machines. The proposed scheme avoids the assumption of fixed metric by implicitly deriving a local distance from the Hessian matrix of a convex function that is used to generate the Bregman distance function. We also present an efficient learning algorithm for the proposed scheme for distance function learning. The extensive experiments with semi-supervised clustering show the proposed technique (i) outperforms the state-of-the-art approaches for distance function learning, and (ii) is computationally efficient for high dimensional data.

## 1 Introduction

An effective distance function plays an important role in many machine learning and data mining techniques. For instance, many clustering algorithms depend on distance functions for the pairwise distance measurements; most information retrieval techniques rely on distance functions to identify the data points that are most similar to a given query; $k$-nearest-neighbor classifier depends on distance functions to identify the nearest neighbors for data classification. In general, learning effective distance functions is a fundamental problem in both data mining and machine learning.

Recently, learning distance functions from data has been actively studied in machine learning. Instead of using a predefined distance function (e.g., Euclidean distance), researchers have attempted to learn distance functions from side information that is often provided in the form of pairwise constraints, i.e., must-link constraints for pairs of similar data points and cannot-link constraints for pairs of dissimilar data points. Example algorithms include [16, 2, 8, 11, 7, 15].

Most distance learning methods assume a Mahalanobis distance. Given two data points $x$ and $x'$, the distance between $x$ and $x'$ is calculated by $d(x, x') = (x - x')^\top A(x - x')$, where $A$ is the distance metric that needs to be learned from the side information. [16] learns a global distance metric (GDM) by minimizing the distance between similar data points while keeping dissimilar data points far apart. It requires solving a Semi-Definite Programming (SDP) problem, which is computationally expensive when the dimensionality is high. BarHillel et al [2] proposed the Relevant Components Analysis (RCA), which is computationally efficient and achieves comparable results as GDM. The main drawback with RCA is that it is unable to handle the cannot-link constraints. This problem was addressed by Discriminative Component Analysis (DCA) in [8], which learns a distance metric by minimizing the distance between similar data points and in the meantime maximizing the distance

between dissimilar data points. The authors in [4] proposed an information-theoretic based metric learning approach (ITML) that learns the Mahalanobis distance by minimizing the differential relative entropy between two multivariate Gaussians. Neighborhood Component Analysis (NCA) [5] learns a distance metric by extending the nearest neighbor classifier. The maximum-margin nearest neighbor (LMNN) classifier [14] extends NCA through a maximum margin framework. Yang et al. [17] propose a Local Distance Metric (LDM) that addresses multimodal data distributions. Hoi et al. [7] propose a semi-supervised distance metric learning approach that explores the unlabeled data for metric learning. In addition to learning a distance metric, several studies [12, 6] are devoted to learning a distance function, mostly non-metric, from the side information.

Despite the success, the existing approaches for distance metric learning are limited in two aspects. First, most existing methods assume a fixed distance metric for the entire input space, which make it difficult for them to handle the heterogeneous data. This issue was already demonstrated in [17] when learning distance metrics from multi-modal data distributions. Second, the existing methods aim to learn a full matrix for the target distance metric that is in the square of the dimensionality, making it computationally unattractive for high dimensional data. Although the computation can be reduced significantly by assuming certain forms of the distance metric (e.g., diagonal matrix), these simplifications often lead to suboptimal solutions. To address these two limitations, we propose a novel scheme that learns Bregman distance functions from the given side information. Bregman distance or Bregman divergence [3] has several salient properties for distance measure. Bregman distance generalizes the class of Mahalanobis distance by deriving a distance function from a given convex function $\phi(x)$. Since the local distance metric can be derived from the local Hessian matrix of $\varphi(x)$, Bregman distance function avoids the assumption of fixed distance metric. Recent studies [1] also reveal the connection between Bregman distances and exponential families of distributions. For example, Kullback-Leibler divergence is a special Bregman distance when choosing the negative entropy function for the convex function $\varphi(x)$.

The objective of this work is to design an efficient and effective algorithm that learns a Bregman distance function from pairwise constraints. Although Bregman distance or Bregman divergence has been explored in [1], all these studies assume a predefined Bregman distance function. To the best of our knowledge, this is the first work that addresses the problem of learning Bregman distances from the pairwise constraints. We present a non-parametric framework for Bregman distance learning, together with an efficient learning algorithm. Our empirical study with semi-supervised clustering show that the proposed approach (i) outperforms the state-of-the-art algorithms for distance metric learning, and (ii) is computationally efficient for high dimensional data.

The rest of the paper is organized as follows. Section 2 presents the proposed framework of learning Bregman distance functions from the pairwise constraints, together with an efficient learning algorithm. Section 3 presents the experimental results with semi-supervised clustering by comparing the proposed algorithms with a number of state-of-the-art algorithms for distance metric learning. Section 5 concludes this work.

## 2 Learning Bregman Distance Functions

### 2.1 Bregman Distance Function

Bregman distance function is defined based on a given convex function. Let $\varphi(x) : \mathbb{R}^d \mapsto \mathbb{R}$ be a strictly convex function that is twice differentiable. Given $\varphi(x)$, the Bregman distance function is defined as

$$d(x_1, x_2) = \varphi(x_1) - \varphi(x_2) - (x_1 - x_2)^\top \nabla \varphi(x_2)$$

For the convenience of discussion, we consider a symmetrized version of the Bregman distance function that is defined as follows

$$d(x_1, x_2) = (\nabla \varphi(x_1) - \nabla \varphi(x_2))^\top (x_1 - x_2) \tag{1}$$

The following proposition shows the properties of $d(x_1, x_2)$.

**Proposition 1.** *The distance function defined in (1) satisfies the following properties if $\varphi(x)$ is a strictly convex function: (a) $d(x_1, x_2) = d(x_2, x_1)$, (b) $d(x_1, x_2) \geq 0$, (c) $d(x_1, x_2) = 0 \leftrightarrow x_1 = x_2$*

**Remark**  To better understand the Bregman distance function, we can rewrite $d(x_1, x_2)$ in (1) as

$$d(x_1, x_2) = (x_1 - x_2)^\top \nabla^2 \varphi(\tilde{x})(x_1 - x_2)$$

where $\tilde{x}$ is a point on the line segment between $x_1$ and $x_2$. As indicated by the above expression, the Bregman distance function can be viewed as a general Mahalanobis distance that introduces a local distance metric $A = \nabla^2\varphi(\tilde{x})$. Unlike the conventional Mahalanobis distance where metric $A$ is a constant matrix throughout the entire space, the local distance metric $A = \nabla^2\varphi(\tilde{x})$ is introduced via the Hessian matrix of convex function $\varphi(x)$ and therefore depends on the location of $x_1$ and $x_2$.

Although the Bregman distance function defined in (1) does not satisfy the triangle inequality, the following proposition shows the degree of violation could be bounded if the Hessian matrix of $\varphi(x)$ is bounded.

**Proposition 2.** *Let $\Omega$ be the closed domain for $x$. If $\exists m, M \in \mathbb{R}$, $M > m > 0$ and*

$$mI \preceq \min_{x\in\Omega} \nabla^2\varphi(x) \preceq \max_{x\in\Omega} \nabla^2\varphi(x) \preceq MI$$

*where $I$ is the identity matrix, we have the following inequality*

$$\sqrt{d(x_a, x_b)} \leq \sqrt{d(x_a, x_c)} + \sqrt{d(x_c, x_b)} + (\sqrt{M} - \sqrt{m})[d(x_a, x_c)d(x_c, x_b)]^{1/4} \qquad (2)$$

The proof of this proposition can be found in Appendix A. As indicated by Proposition 2, the degree of violation of the triangle inequality is essentially controlled by $\sqrt{M} - \sqrt{m}$. Given a smooth convex function with almost constant Hessian matrix, we would expect that to a large degree, Bregman distance will satisfy the triangle inequality. In the extreme case when $\varphi(x) = x^\top A x/2$ and $\nabla^2\varphi(x) = A$, we have a constant Hessian matrix, leading to a complete satisfaction of the triangle inequality.

## 2.2 Problem Formulation

To a learn a Bregman distance function, the key is to find the appropriate convex function $\varphi(x)$ that is consistent with the given pairwise constraints. In order to learn the convex function $\varphi(x)$, we take a non-parametric approach by assuming that $\varphi(\cdot)$ belongs to a Reproducing Kernel Hilbert Space $\mathcal{H}_\kappa$. Given a kernel function $\kappa(x, x') : \mathbb{R}^d \times \mathbb{R}^d \mapsto \mathbb{R}$, our goal is to search for a convex function $\varphi(x) \in \mathcal{H}_\kappa$ such that the induced Bregman distance function, denoted by $d_\varphi(x, x')$, minimizes the overall training error with respect to the given pairwise constraints.

We denote by $\mathcal{D} = \{(x_i^1, x_i^2, y_i), i = 1, \ldots, n\}$ the collection of pairwise constraints for training. Each pairwise constraint consists of a pair of instances $x_i^1$ and $x_i^2$, and a label $y_i$ that is $+1$ if $x_i^1$ and $x_i^2$ are similar and $-1$ if $x_i^1$ and $x_i^2$ are dissimilar. We also introduce $X = (x_1, \ldots, x_N)$ to include the input patterns of all training instances in $\mathcal{D}$.

Following the maximum margin framework for classification, we cast the problem of learning a Bregman distance function from pairwise constraints into the following optimization problem, i.e.,

$$\min_{\varphi\in\Omega(\mathcal{H}_\kappa), b\in\mathbb{R}_+} \quad \frac{1}{2}|\varphi|^2_{\mathcal{H}_\kappa} + C\sum_{i=1}^{n} \ell(y_i[d(x_i^1, x_i^2) - b]) \qquad (3)$$

where $\Omega(\mathcal{H}) = \{f \in \mathcal{H} : f \text{ is convex}\}$ refers to the subspace of functional space $\mathcal{H}$ that only includes convex functions, $\ell(z) = \max(0, 1 - z)$ is a hinge loss, and $C$ is a penalty cost parameter.

The main challenge with solving the variational problem in (3) is that it is difficult to derive a represerter theorem for $\varphi(x)$ because it is $\nabla\varphi(x)$ used in the definition of distance function, not $\varphi(x)$. Note that although it seems to be convenient to regularize $\nabla\varphi(x)$, it will be difficult to restrict $\varphi(x)$ to be convex. To resolve this problem, we consider a special family of kernel functions $\kappa(x, x')$ that has the form $\kappa(x_1, x_2) = h(x_1^\top x_2)$ where $h : \mathbb{R} \mapsto \mathbb{R}$ is a strictly convex function. Examples of $h(z)$ that guarantees $\kappa(\cdot, \cdot)$ to be positive semi-definite are $h(z) = |z|^d$ ($d \geq 1$), $h(z) = |z+1|^d$ ($d \geq 1$), and $h(z) = \exp(z)$. For the convenience of discussion, we assume $h(0) = 0$ throughout this paper.

First, since $\varphi(x) \in \mathcal{H}_\kappa$, we have

$$\varphi(x) = \int dy \kappa(x, y) q(y) = \int dy h(x^\top y) q(y) \qquad (4)$$

where $q(y)$ is a weighting function. Given the training instances $x_1, \ldots, x_N$, we divide the space $\mathbb{R}^d$ into $\mathcal{A}$ and $\bar{\mathcal{A}}$ that are defined as

$$\mathcal{A} = \text{span}\{x_1, \ldots, x_N\}, \ \bar{\mathcal{A}} = \text{Null}(x_1, \ldots, x_N) \qquad (5)$$

We define $\mathcal{H}_{\parallel}$ and $\mathcal{H}_{\perp}$ as follows

$$\mathcal{H}_{\parallel} = \mathrm{span}\{\kappa(x, \cdot), \forall x \in \mathcal{A}\}, \ \mathcal{H}_{\perp} = \mathrm{span}\{\kappa(x, \cdot), \forall x \in \bar{\mathcal{A}}\} \tag{6}$$

The following proposition summarizes an important property of reproducing kernel Hilbert space $\mathcal{H}_{\kappa}$ when kernel function $\kappa(\cdot, \cdot)$ is restricted to the form in Eq. (2.2).

**Proposition 3.** *If the kernel function $\kappa(\cdot, \cdot)$ is written in the form of Equation (2.2) with $h(0) = 0$, we have $\mathcal{H}_{\parallel}$ and $\mathcal{H}_{\perp}$ form a complete partition of $\mathcal{H}_{\kappa}$, i.e., $\mathcal{H}_{\kappa} = \mathcal{H}_{\parallel} \cup \mathcal{H}_{\perp}$, and $\mathcal{H}_{\parallel} \perp \mathcal{H}_{\perp}$.*

We therefore have the following representer theorem for $\varphi(x)$ that minimizes (3)

**Theorem 1.** *The function $\varphi(x)$ that minimizes (3) admits the following expression*

$$\varphi(x) \in \mathcal{H}_{\parallel} = \int_{y \in \mathcal{A}} dy q(y) h(x^\top y) = \int du q(u) h(x^\top X u) \tag{7}$$

*where $u \in \mathbb{R}^N$ and $X = (x_1, \ldots, x_N)$.*

The proof of the above theorem can be found in Appendix B.

## 2.3  Algorithm

To further derive a concrete expression for $\varphi(x)$, we restrict $q(y)$ in (7) to the special form: $q(y) = \sum_{i=1}^N \alpha_i \delta(y - x_i)$ where $\alpha_i \geq 0, i = 1, \ldots, N$ are non-negative combination weights. This results in $\varphi(x) = \sum_{i=1}^N \alpha_i h(x_i^\top x)$, and consequently $d(x_a, x_b)$ as follows

$$d(x_a, x_b) = \sum_{i=1}^N \alpha_i (h'(x_a^\top x_i) - h'(x_b^\top x_i)) x_i^\top (x_a - x_b) \tag{8}$$

By defining $h(x_a) = (h'(x_a^\top x_1), \ldots, h'(x_a^\top x_N))^\top$, we can express $d(x_a, x_b)$ as follows

$$d(x_a, x_b) = (x_a - x_b)^\top X (\alpha \circ [h(x_a) - h(x_b)]) \tag{9}$$

Notice that when $h(z) = z^2/2$, we have $d(x_a, x_b)$ expressed as

$$d(x_a, x_b) = (x_a - x_b)^\top X \mathrm{diag}(\alpha) X^\top (x_a - x_b). \tag{10}$$

This is a Mahanalobis distance with metric $A = X \mathrm{diag}(\alpha) X^\top = \sum_{i=1}^N \alpha_i x_i x_i^\top$. When $h(z) = \exp(z)$, we have $h(x) = (\exp(x^\top x_1), \ldots, \exp(x^\top x_N))$, and the resulting distance function is no longer stationary due to the non-linear function $\exp(z)$.

Given the assumption that $q(y) = \sum_{i=1}^N \alpha_i \delta(y - x_i)$, we have (3) simplified as

$$\min_{\alpha \in \mathbb{R}^N, b} \ \frac{1}{2} \alpha^\top K \alpha + C \sum_{i=1}^n \varepsilon_i \tag{11}$$

$$\text{s. t.} \quad y_i \left((x_i^1 - x_i^2)^\top X (\alpha \circ [h(x_i^1) - h(x_i^2)]) - b\right) \geq 1 - \varepsilon_i,$$
$$\varepsilon_i \geq 0, \ i = 1, \ldots, n, \alpha_k \geq 0, k = 1, \ldots, N$$

Note that the constraint $\alpha_k \geq 0$ is introduced to ensure $\varphi(x) = \sum_{k=1}^N \alpha_k h(x^\top x_k)$ is a convex function. By defining

$$z_i = [h(x_i^1) - h(x_i^2)] \circ [X^\top (x_i^1 - x_i^2)], \tag{12}$$

we simplify the problem in (11) as follows

$$\min_{\alpha \in \mathbb{R}_+^N, b} \quad \mathcal{L} = \frac{1}{2} \alpha^\top K \alpha + C \sum_{i=1}^n \ell(y_i[z_i^\top \alpha - b]) \tag{13}$$

where $\ell(z) = \max(0, 1 - z)$.

We solve the above problem by a simple subgradient descent approach. In particular, at iteration $t$, given the current solution $\alpha^t$ and $b^t$, we compute the gradients as

$$\nabla_\alpha \mathcal{L} = K\alpha^t + C\sum_{i=1}^n \partial\ell(y_i[z_i^\top \alpha^t - b^t])y_i z_i, \ \nabla_b \mathcal{L} = -C\sum_{i=1}^n \partial\ell(y_i[z_i^\top \alpha^t - b^t])y_i \qquad (14)$$

where $\partial\ell(z)$ stands for the subgradient of $\ell(z)$. Let $\mathcal{S}_t^+ \in \mathcal{D}$ denotes the set of training instances for which $(\alpha^t, b^t)$ suffers a non-zeros loss, i.e.,

$$\mathcal{S}_t^+ = \{(z_i, y_i) \in \mathcal{D} : y_i(z_i^\top \alpha^t - b^t) < 1\} \qquad (15)$$

We can then express the sub-gradients of $\mathcal{L}$ at $\alpha^t$ and $b^t$ as follows:

$$\nabla_\alpha \mathcal{L} = K\alpha - C\sum_{(z_i, y_i) \in \mathcal{S}_t^+} y_i z_i, \ \nabla_b \mathcal{L} = C\sum_{(z_i, y_i) \in \mathcal{S}_t^+} y_i \qquad (16)$$

The new solution, denoted by $\alpha^{t+1}$ and $b^{t+1}$, is computed as follows:

$$\alpha_k^{t+1} = \pi_{[0, +\infty]}\left(\alpha_k^t - \gamma_t[\nabla_\alpha \mathcal{L}]_k\right), \ b^{t+1} = b^t - \gamma_t \nabla_b \mathcal{L} \qquad (17)$$

where $\alpha_k^{t+1}$ is the $k$-th element of vector $\alpha^{t+1}$, $\pi_G(x)$ projects $x$ into the domain $G$, and $\gamma_t$ is the step size that is set to be $\gamma_t = \frac{C}{t}$ by following the Pegasos algorithm [10] for solving SVMs. The pseudo-code of the proposed algorithm is summarized in Algorithm 1.

---

**Algorithm 1** Algorithm of Learning Bregman Distance Functions

---

INPUT:
- data matrix: $X \in \mathbb{R}^{N \times d}$
- pair-wise constraint $\{(x_i^1, x_i^2, y^i), i = 1, \dots, n\}$
- kernel function: $\kappa(x_1, x_2) = h(x_1^\top x_2)$
- penalty cost parameter $C$

OUTPUT:
- Bregman coefficients $\alpha \in \mathbb{R}_+^N, b \in \mathbb{R}$

PROCEDURE

1: initialize Bregman coefficients: $\alpha = \alpha_0, b = b_0$
2: calculate kernel matrix: $K = [h(x_i^\top x_j)]_{N \times N}$
3: calculate vectors $z_i$: $z_i = [h(x_i^1) - h(x_i^2)] \circ [X^\top(x_i^1 - x_i^2)]$
4: set iteration step $t = 1$;
5: **repeat**
6:     (1) update the learning rate: $\gamma = C/t, t = t + 1$
7:     (2) update subset of training instances: $\mathcal{S}_t^+ = \{(z_i, y_i) \in \mathcal{D} : y_i(z_i^\top \alpha - b) < 1\}$
8:     (3) compute the gradients w.r.t $\alpha$ and $b$:
9:         $\nabla_\alpha \mathcal{L} = K\alpha - C\sum_{z_i \in \mathcal{S}_t^+} y_i z_i, \nabla_b \mathcal{L} = C\sum_{z_i \in \mathcal{S}_t^+} y_i$
10:    (4) update Bregman coefficients $\alpha = (\alpha_1, \dots, \alpha_n)$ and threshold $b$:
11:       $b \leftarrow b - \gamma \nabla_b \mathcal{L}, \alpha_k \leftarrow \pi_{[0, +\infty]}\left(\alpha_k - \gamma[\nabla_\alpha \mathcal{L}]_k\right), k = 1, \dots, N$
12: **until** convergence

---

**Computational complexity** One of the major computational costs for Algorithm 1 is the preparation of kernel matrix $K$ and vector $\{z_i\}_{i=1}^n$, which fortunately can be pre-computed. Each step of the subgradient descent algorithm has a linear complexity, i.e., $\mathcal{O}(\max(N, n))$, which makes it reasonable even for large data sets with high dimensionality. The number of iterations for convergence is $O(1/\epsilon^2)$ where $\epsilon$ is the target accuracy. It thus works fine if we are not critical about the accuracy of the solution.

## 3 Experiments

We evaluate the proposed distance learning technique by semi-supervised clustering. In particular, we first learn a distance function from the given pairwise constraints and then apply the learned distance function to data clustering. We verify the efficacy and efficiency of the proposed technique by comparing it with a number of state-of-the-art algorithms for distance metric learning.

### 3.1 Experimental Testbed and Settings

We adopt six well-known datasets from UCI machine learning repository, and six popular text benchmark datasets[1] in our experiments. These datasets are chosen for clustering because they vary signif-

icantly in properties such as the number of clusters/classes, the number of features, and the number of instances. The diversity of datasets allows us to examine the effectiveness of the proposed learning technique more comprehensively. The details of the datasets are shown in Table 1.

| dataset | #samples | #feature | #classes | dataset | #samples | #feature | #classes |
|---|---|---|---|---|---|---|---|
| breast-cancer | 683 | 10 | 2 | w1a | 2,477 | 300 | 2 |
| diabetes | 768 | 8 | 2 | w2a | 3,470 | 300 | 2 |
| ionosphere | 251 | 34 | 2 | w6a | 17,188 | 300 | 2 |
| liver-disorders | 345 | 6 | 2 | WebKB | 4,291 | 19,687 | 6 |
| sonar | 208 | 60 | 2 | newsgroup | 7,149 | 47,411 | 11 |
| a1a | 1,605 | 123 | 2 | Reuter | 10,789 | 5,189 | 79 |

Table 1: The details of our experimental testbed

Similar to previous work [16], the pairwise constraints are created by random sampling. More specifically, we randomly sample a subset of pairs from the pool of all possible pairs (every two instances forms a pair). Two instances form a must-link constraint (i.e., $y_i = +1$) if they share the same class label, and form a cannot-link constraint (i.e., $y_i = -1$) if they are assigned to different classes. To calculate the Bregman function, in this experiment, we adopt the non-linear function $h(x) = (\exp(x^\top x_1), \ldots, \exp(x^\top x_N))$.

To perform data clustering, we run the k-means algorithm using the distance function learned from 500 randomly sampled positive constraints 500 random negative constraints. The number of clusters is simply set to the number of classes in the ground truth. The initial cluster centroids are randomly chosen from the dataset. To enable fair comparisons, all comparing algorithms start with the same set of initial centroids. We repeat each clustering experiment for 20 times, and report the final results by averaging over the 20 runs.

We compare the proposed Bregman distance learning method using the k-means algorithm for semi-supervised clustering, termed **Bk-means**, with the following approaches: (1) a standard *k-means*, (2) the constrained k-means [13] (*Ck-means*), (3) Ck-means with distance learned by *RCA* [2], (4) Ck-means with distance learned by *DCA* [8], (5) Ck-means with distance learned by the Xing's algorithm [16] (*Xing*), (6) Ck-means with information-theoretic metric learning (*ITML*) [4], and (7) Ck-means with a distance function learned by a boosting algorithm (*DistBoost*) [12].

To evaluate the clustering performance, we use the some standard performance metrics, including pairwise Precision, pairwise Recall, and pairwise F1 measures [9], which are evaluated base on the pairwise results. Specifically, *pairwise precision* is the ratio of the number of *correct* pairs placed in the same cluster over the total number of pairs placed in the same cluster, *pairwise recall* is the ratio of the number of correct pairs placed in the same cluster over the total number of pairs *actually* placed in the same cluster, and *pairwise F1* equals to $2 \times precision \times recall/(precision + recall)$.

### 3.2 Performance Evaluation on Low-dimensional Datasets

The first set of experiments evaluates the clustering performance on six UCI datasets. Table 2 shows the average precision, recall, and F1 measurements of all the competing algorithms given a set of $1,000$ random constraints. The top two highest average F1 scores on each dataset were highlighted in **bold** font. From the results in Table 2, we observe that the proposed Bregman distance based k-means clustering approach (Bk-means) is either the best or the second best for almost all datasets, indicating that the proposed algorithm is in general more effective than the other algorithms for distance metric learning.

### 3.3 Performance Evaluation on High-dimensional Text Data

We evaluate the clustering performance on six text datasets. Since some of the methods are infeasible for text clustering due to the high dimensionality, we only include the results for the methods which are feasible for this experiment (i.e., OOM indicates the method takes more than 10 hours, and OOT indicates the method needs more than 16G REM). Table 3 summarizes the F1 performance of all feasible methods for datasets w1a, w2a, w6a, WebKB, 20newsgroup and reuter. Since cosine similarity is commonly used in textual domain, we use k-means, Ck-means in both Euclidian space and Cosine similarity space as baselines. The best F1 scores are marked in **bold** in Table 3. The results show that the learned Bregman distance function is applicable for high dimensional data, and it outperforms the other commonly used text clustering methods for four out of six datasets.

| method | breast | | | diabetes | | |
|---|---|---|---|---|---|---|
| | precision | recall | F1 | precision | recall | F1 |
| baseline | 72.85±3.77 | 72.52±2.30 | 72.73±3.42 | 52.47±8.93 | 57.17±3.68 | 56.41±4.53 |
| Ck-means | 98.10±2.20 | 81.01±0.10 | 85.31±1.48 | 60.06±1.13 | 55.98±0.64 | 57.57±0.85 |
| ITML | 97.05±2.77 | 88.96±0.30 | 91.94±2.15 | 73.93±1.28 | 70.11±0.41 | **71.55**±0.81 |
| Xing | 93.61±0.14 | 84.19±0.83 | 88.11±0.22 | 58.11±0.48 | 58.31±0.16 | 58.21±0.31 |
| RCA | 85.40±0.14 | 94.16±0.29 | 90.18±2.94 | 59.86±2.99 | 62.70±2.18 | 61.22±2.59 |
| DCA | 94.53±0.34 | 93.23±0.29 | **93.88**±0.22 | 61.23±2.05 | 64.88±0.56 | 63.00±0.75 |
| DistBoost | 94.76±0.24 | 93.83±0.31 | 94.29±0.29 | 64.45±1.02 | 68.33±0.98 | 66.33±1.00 |
| Bk-means | 99.04±0.10 | 98.33±0.24 | **98.37**±0.19 | 99.42±0.40 | 64.68±0.63 | **77.43**±0.92 |

| method | ionosphere | | | liver-disorders | | |
|---|---|---|---|---|---|---|
| | precision | recall | F1 | precision | recall | F1 |
| baseline | 62.35±6.30 | 53.39±2.74 | 57.28±6.20 | 63.92±8.60 | 50.50±0.40 | 55.67±5.96 |
| Ck-means | 57.05±1.24 | 51.28±1.58 | 61.46±1.36 | 62.90±8.43 | 50.35±1.68 | 55.13±1.63 |
| ITML | 97.10±2.70 | 59.99±0.31 | 72.62±1.24 | 93.53±3.28 | 55.57±0.10 | **68.73**±1.40 |
| Xing | 63.46±0.11 | 64.10±0.03 | 63.52±0.39 | 95.42±2.85 | 49.65±0.08 | 65.31±1.10 |
| RCA | 100.00±6.19 | 50.36±1.44 | 66.99±0.45 | 59.56±18.95 | 52.15±1.68 | 54.92±5.76 |
| DCA | 66.36±3.01 | 67.01±2.12 | 66.68±0.00 | 70.18±4.27 | 50.41±0.07 | 58.67±1.63 |
| DistBoost | 75.91±1.11 | 69.34±0.91 | **72.72**±1.03 | 51.60±1.43 | 52.88±1.31 | 52.23±1.37 |
| Bk-means | 97.64±1.93 | 62.71±1.94 | **73.28**±1.93 | 96.89±4.11 | 50.29±2.09 | **66.86**±3.10 |

| method | sonar | | | a1a | | |
|---|---|---|---|---|---|---|
| | precision | recall | F1 | precision | recall | F1 |
| baseline | 52.98±2.05 | 50.84±1.69 | 51.87±1.47 | 55.81±1.01 | 69.99±0.91 | 62.10±0.99 |
| Ck-means | 60.44±4.53 | 51.71±1.17 | 55.32±1.37 | 69.91±0.08 | 80.34±0.18 | 77.01±0.12 |
| ITML | 98.68±2.46 | 56.31±2.28 | 70.46±2.35 | 99.99±0.98 | 70.30±0.54 | **81.76**±0.76 |
| Xing | 96.99±4.53 | 69.81±0.05 | **79.83**±2.70 | 57.70±1.32 | 70.89±1.01 | 63.62±1.21 |
| RCA | 100.00±13.69 | 69.81±1.33 | **79.83**±5.85 | 76.64±0.08 | 66.96±0.35 | 69.96±0.18 |
| DCA | 100.00±0.64 | 59.75±0.30 | 73.11±0.57 | 57.15±1.32 | 71.76±1.87 | 63.63±1.55 |
| DistBoost | 76.64±0.57 | 74.48±0.69 | 75.54±0.62 | n/a | n/a | n/a |
| Bk-means | 99.20±1.62 | 74.24±1.23 | **82.52**±1.44 | 99.98±0.21 | 77.72±0.17 | **86.32**±0.19 |

Table 2: Evaluation of clustering performance (average precision, recall, and F1) on six UCI datasets. The top two F1 scores are highlighted in bold font for each dataset.

| methods | w1a | w2a | w6a | WebKB | newsgroup | Reuter |
|---|---|---|---|---|---|---|
| k-means(EU) | 76.68±0.25 | 72.59±0.77 | 76.52±0.97 | 35.78±0.17 | 16.54±0.05 | 43.88±0.23 |
| k-means(Cos) | 76.87±5.61 | 73.47±1.35 | 77.16±1.27 | 35.18±3.41 | 18.87±0.14 | 45.42±0.73 |
| Ck-means(EU) | 87.04±1.15 | **97.23**±1.21 | 76.52±1.01 | 70.84±2.29 | 19.12±0.54 | 56.00±0.42 |
| Ck-means(Cos) | 87.14±2.14 | 97.14±2.12 | 75.32±0.91 | **75.84**±1.08 | 20.08±0.49 | 58.24±0.82 |
| RCA | 91.00±1.02 | 96.45±1.17 | 93.51±1.13 | OOM | OOM | OOT |
| DCA | 92.13±1.04 | 94.30±2.56 | 87.44±1.99 | OOM | OOM | OOT |
| ITML | 92.31±0.84 | 94.12±0.92 | 96.95 ±0.13 | OOT | OOM | OOT |
| Bk-means | **93.43**±1.07 | 96.92±1.02 | **98.64**±0.24 | 73.94±1.25 | **25.17**±1.27 | **64.51**±0.95 |

Table 3: Evaluation of clustering F1 performance on the high dimensional text data. Only applicable methods are shown. OOM indicates "out of memory", and OOT indicates "out of time".

### 3.4 Computational Complexity

Here, we evaluate the running time of semi-supervised clustering. For a conventional clustering algorithm such as k-means, its computational complexity is determined by both the calculation of distance and the clustering scheme. For a semi-supervised clustering algorithm based on distance learning, the overall computational time include both the time for training an appropriate distance function and the time for clustering data points. The average running times of semi-supervised clustering over the six UCI datasets are listed in Table 4. It is clear that the Bregman distance based clustering has comparable efficiency with simple methods like RCA and DCA on low dimensional data, and runs much faster than Xing, ITML, and DistBoost. On the high dimensional text data, it is much faster than other applicable DML methods.

| Algorithm | k-means | Ck-means | ITML | Xing | RCA | DCA | DistBoost | Bk-means |
|---|---|---|---|---|---|---|---|---|
| UCI data(Sec.) | 0.51 | 0.72 | 7.59 | 8.56 | 0.88 | 0.90 | 13.09 | 1.70 |
| Text data(Min.) | 0.78 | 4.56 | 71.55 | n/a | 68.90 | 69.34 | n/a | 3.84 |

Table 4: Comparison of average running time over the six UCI datasets and subsets of six text datasets (10% sampling from the datasets in Table 1).

# 4  Conclusions

In this paper, we propose to learn a Bregman distance function for clustering algorithms using a non-parametric approach. The proposed scheme explicitly address two shortcomings of the existing approaches for distance fuction/metric learning, i.e., assuming a fixed distance metric for the entire input space and high computational cost for high dimensional data. We incorporate the Bregman distance function into the k-means clustering algorithm for semi-supervised data clustering. Experiments of semi-supervised clustering with six UCI datasets and six high dimensional text datasets have shown that the Bregman distance function outperforms other distance metric learning algorithms in F1 measure. It also verifies that the proposed distance learning algorithm is computationally efficient, and is capable of handling high dimensional data.

## Acknowledgements

This work was done when Mr. Lei Wu was an RA at Nanyang Technological University, Singapore. This work was supported in part by MOE tier-1 Grant (RG67/07), NRF IDM Grant (NRF2008IDM-IDM-004-018), National Science Foundation (IIS-0643494), and US Navy Research Office (N00014-09-1-0663).

## APPENDIX A: Proof of Proposition 2

*Proof.* First, let us denote by $f$ as follows:

$$f = (\sqrt{M} - \sqrt{m})[d(x_a, x_c)d(x_c, x_b)]^{1/4}$$

The square of the right side of Eq. (2) is

$$(\sqrt{d(x_a, x_c)} + \sqrt{d(x_c, x_b)} + f^{1/4})^2 = d(x_a, x_b) - \eta(x_a, x_b, x_c) + \delta(x_a, x_b, x_c)$$

where

$$
\begin{aligned}
\delta(x_a, x_b, x_c) &= f^2 + 2f\sqrt{d(x_a, x_c)} + 2f\sqrt{d(x_c, x_b)} + 2\sqrt{d(x_a, x_c)d(x_c, x_b)} \\
\eta(x_a, x_b, x_c) &= (\nabla\varphi(x_a) - \nabla\varphi(x_c))(x_c - x_b) + (\nabla\varphi(x_c) - \nabla\varphi(x_b))(x_a - x_c).
\end{aligned}
$$

From this above equation, the proposition holds if and only if $\delta(x_a, x_b, x_c) - \eta(x_a, x_b, x_c) \geq 0$. From the fact that

$$
\begin{aligned}
&\delta(x_a, x_b, x_c) - \eta(x_a, x_b, x_c) \\
&= \frac{(\sqrt{M} - \sqrt{m})^2 + 2(\sqrt{M} - \sqrt{m})\left(d(x_a, x_c)^{\frac{3}{4}}d(x_c, x_b)^{\frac{1}{4}} + d(x_c, x_b)^{\frac{3}{4}}d(x_a, x_c)^{\frac{1}{4}}\right) + 2d(x_a, x_c)d(x_c, x_b)}{\sqrt{d(x_a, x_c)d(x_c, x_b)}}
\end{aligned}
$$

since $\sqrt{M} > \sqrt{m}$ and the distance function $d(\cdot) \geq 0$, we get $\delta(x_a, x_b, x_c) - \eta(x_a, x_b, x_c) \geq 0$. □

## APPENDIX B: Proof of Theorem 1

*Proof.* We write $\varphi(x) = \varphi_{\parallel}(x) + \varphi_{\perp}(x)$ where

$$\varphi_{\parallel}(x) \in \mathcal{H}_{\parallel} = \int_{y\in\mathcal{A}} dy q(y) h(x^\top y), \ \varphi_{\perp}(x) \in \mathcal{H}_{\perp} = \int_{y\in\bar{\mathcal{A}}} dy q(y) h(x^\top y)$$

Thus, the distance function defined in (1) is then expressed as

$$
\begin{aligned}
d(x_a, x_b) &= (x_a - x_b)^\top \left(\nabla\varphi_{\parallel}(x_a) - \nabla\varphi_{\parallel}(x_b)\right) + (x_a - x_b)^\top \left(\nabla\varphi_{\perp}(x_a) - \nabla\varphi_{\perp}(x_b)\right) \\
&= \int_{y\in\mathcal{A}} q(y)(h'(x_a^\top y) - h'(x_b^\top y))y^\top (x_a - x_b) + \int_{y\in\bar{\mathcal{A}}} q(y)(h'(x_a^\top y) - h'(x_b^\top y))y^\top (x_a - x_b) \\
&= \int_{y\in\mathcal{A}} q(y)(h'(x_a^\top y) - h'(x_b^\top y))y^\top (x_a - x_b) = (x_a - x_b)^\top \left(\nabla\varphi_{\parallel}(x_a) - \nabla\varphi_{\parallel}(x_b)\right)
\end{aligned}
$$

Since $|\varphi(x)|_{\mathcal{H}_\kappa}^2 = |\varphi_{\parallel}(x)|_{\mathcal{H}_\kappa}^2 + |\varphi_{\perp}(x)|_{\mathcal{H}_\kappa}^2$, the minimizer of (1) should have $|\varphi_{\perp}(x)|_{\mathcal{H}_\kappa}^2 = 0$. Since $|\varphi_{\perp}(x)| = \langle\varphi_{\perp}(\cdot), \kappa(x,\cdot)\rangle_{\mathcal{H}_\kappa} \leq |\kappa(x,\cdot)|_{\mathcal{H}_\kappa}|\varphi_{\perp}|_{\mathcal{H}_\kappa} = 0,$, we have $\varphi_{\perp}(x) = 0$ for any $x$. We thus have $\varphi(x) = \varphi_{\parallel}(x)$, which leads to the result in the theorem. □

## Footnotes

[1]The Reuter dataset is available at: http://renatocorrea.googlepages.com/textcategorizationdatasets

# References

[1] A. Banerjee, S. Merugu, I. Dhillon, and J. Ghosh. Clustering with bregman divergences. In *Journal of Machine Learning Research*, pages 234–245, 2004.

[2] A. Bar-Hillel, T. Hertz, N. Shental, and D. Weinshall. Learning a mahalanobis metric from equivalence constraints. *JMLR*, 6:937–965, 2005.

[3] L. Bregman. The relaxation method of finding the common points of convex sets and its application to the solution of problems in convex programming. *USSR Computational Mathematics and Mathematical Physics*, 7:200–217, 1967.

[4] J. V. Davis, B. Kulis, P. Jain, S. Sra, and I. S. Dhillon. Information-theoretic metric learning. In *ICML'07*, pages 209–216, Corvalis, Oregon, 2007.

[5] J. Goldberger, S. Roweis, G. Hinton, and R. Salakhutdinov. Neighborhood component analysis. In *NIPS*.

[6] T. Hertz, A. B. Hillel, and D. Weinshall. Learning a kernel function for classification with small training samples. In *ICML '06: Proceedings of the 23rd international conference on Machine learning*, pages 401–408. ACM, 2006.

[7] S. C. H. Hoi, W. Liu, and S.-F. Chang. Semi-supervised distance metric learning for collaborative image retrieval. In *Proceedings of IEEE Conference on Computer Vision and Pattern Recognition (CVPR2008)*, June 2008.

[8] S. C. H. Hoi, W. Liu, M. R. Lyu, and W.-Y. Ma. Learning distance metrics with contextual constraints for image retrieval. In *Proc. CVPR2006*, New York, US, June 17–22 2006.

[9] Y. Liu, R. Jin, and A. K. Jain. Boostcluster: boosting clustering by pairwise constraints. In *KDD'07*, pages 450–459, San Jose, California, USA, 2007.

[10] S. Shalev-Shwartz, Y. Singer, and N. Srebro. Pegasos: Primal estimated sub-gradient solver for svm. In *ICML '07: Proceedings of the 24th international conference on Machine learning*, pages 807–814, New York, NY, USA, 2007. ACM.

[11] L. Si, R. Jin, S. C. H. Hoi, and M. R. Lyu. Collaborative image retrieval via regularized metric learning. *ACM Multimedia Systems Journal*, 12(1):34–44, 2006.

[12] T. H. Tomboy, A. Bar-hillel, and D. Weinshall. Boosting margin based distance functions for clustering. In *In Proceedings of the Twenty-First International Conference on Machine Learning*, pages 393–400, 2004.

[13] K. Wagstaff, C. Cardie, S. Rogers, and S. Schrödl. Constrained k-means clustering with background knowledge. In *ICML'01*, pages 577–584, San Francisco, CA, USA, 2001. Morgan Kaufmann Publishers Inc.

[14] K. Weinberger, J. Blitzer, and L. Saul. Distance metric learning for large margin nearest neighbor classification. In *NIPS 18*, pages 1473–1480, 2006.

[15] L. Wu, S. C. H. Hoi, J. Zhu, R. Jin, and N. Yu. Distance metric learning from uncertain side information with application to automated photo tagging. In *Proceedings of ACM International Conference on Multimedia (MM2009)*, Beijing, China, Oct. 19–24 2009.

[16] E. P. Xing, A. Y. Ng, M. I. Jordan, and S. Russell. Distance metric learning with application to clustering with side-information. In *NIPS2002*, 2002.

[17] L. Yang, R. Jin, R. Sukthankar, and Y. Liu. An efficient algorithm for local distance metric learning. In *Proceedings of the Twenty-Second Conference on Artificial Intelligence (AAAI)*, 2006.

